# Bilinear classifiers for visual recognition

**Hamed Pirsiavash**    **Deva Ramanan**    **Charless Fowlkes**
Department of Computer Science
University of California at Irvine
{hpirsiav,dramanan,fowlkes}@ics.uci.edu

## Abstract

We describe an algorithm for learning bilinear SVMs. Bilinear classifiers are a discriminative variant of bilinear models, which capture the dependence of data on multiple factors. Such models are particularly appropriate for visual data that is better represented as a matrix or tensor, rather than a vector. Matrix encodings allow for more natural regularization through rank restriction. For example, a rank-one scanning-window classifier yields a separable filter. Low-rank models have fewer parameters and so are easier to regularize and faster to score at run-time. We learn low-rank models with bilinear classifiers. We also use bilinear classifiers for transfer learning by sharing linear factors between different classification tasks. Bilinear classifiers are trained with biconvex programs. Such programs are optimized with coordinate descent, where each coordinate step requires solving a convex program - in our case, we use a standard off-the-shelf SVM solver. We demonstrate bilinear SVMs on difficult problems of people detection in video sequences and action classification of video sequences, achieving state-of-the-art results in both.

## 1   Introduction

Linear classifiers (i.e., $w^T x > 0$) are the basic building block of statistical prediction. Though quite standard, they produce many competitive approaches for various prediction tasks. We focus here on the task of visual recognition in video - "does this spatiotemporal window contain an object"? In this domain, scanning-window templates trained with linear classification yield state of the art performance on many benchmark datasets [6, 10, 7].

Bilinear models, introduced into the vision community by [23], provide an interesting generalization of linear models. Here, data points are modelled as the confluence of a pair of factors. Typical examples include digits affected by style and content factors or faces affected by pose and illumination factors. Conditioned on one factor, the model is linear in the other. More generally, one can define multilinear models [25] that are linear in one factor conditioned on the others.

Inspired by the success of bilinear models in data modeling, we introduce discriminative bilinear models for classification. We describe a method for training bilinear (multilinear) SVMs with biconvex (multiconvex) programs. A function $f : X \times Y \to \mathbb{R}$ is called **biconvex** if $f(x,y)$ is convex in $y$ for fixed $x \in X$ and is convex in $x$ for fixed $y \in Y$. Such functions are well-studied in the optimization literature [1, 14]. While not convex, they admit efficient coordinate descent algorithms that solve a convex program at each step. We show bilinear SVM classifiers can be optimized with an off-the-shelf linear SVM solver. This is advantageous because we can leverage large-scale, highly-tuned solvers (we use [13]) to learn bilinear classifiers with tens of thousands of features with hundreds of millions of examples.

While bilinear models are often motivated from the perspective of increasing the flexibility of a linear model, our motivation is reversed - we use them to reduce the number of parameters of a

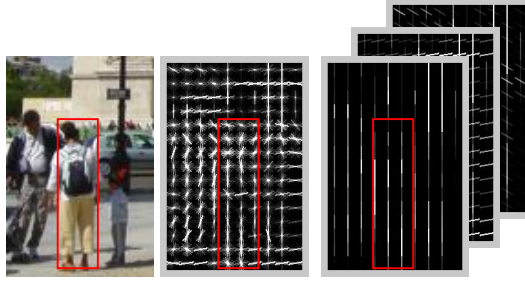

Figure 1: Many approaches for visual recognition employ linear classifiers on scanned windows. Here we illustrate windows processed into gradient-based features [6, 12]. We show an image window (**left**) and a visualization of the extracted HOG descriptor (**middle**), which itself is better represented as gradient features extracted from different orientation channels (**right**). Most learning formulations ignore this natural representation of visual data as matrices or tensors. Wolf et al. [26] show that one can produce more meaningful schemes for regularization and parameter reduction through low-rank approximations of a tensor model. Our contribution involves casting the resulting learning problem as a biconvex optimization. Such formulations can leverage off-the-shelf solvers in an efficient two-stage optimization. We also demonstrate that bilinear models have additional advantages for transfer learning and run-time efficiency.

weight vector that is naturally represented as a matrix or tensor $W$. We reduce parameters by factorizing $W$ into a product of low-rank factors. This parameter reduction can reduce over-fitting and improve run-time efficiency because fewer operations are needed to score an example. These are important considerations when training large-scale spatial or spatiotemporal template-classifiers. In our case, the state-of-the-art features we use to detect pedestrians are based on histograms of gradient (HOG) features [6] or spatio-temporal generalizations [7] as shown in Fig.1. The extracted feature set of both gradient and optical flow histogram is quite large, motivating the need for dimensionality reduction.

Finally, by sharing factors across different classification problems, we introduce a novel formulation of **transfer learning**. We believe that transfer through shared factors is an important benefit of multilinear classifiers which can help ameliorate overfitting.

We begin with a discussion of related work in Sec.2. We then explicitly define our bilinear classifier in Sec. 3. We illustrate several applications and motivations for the bilinear framework in Sec. 4. In Sec. 5, We describe extensions to our model for the multilinear and multiclass case. We provide several experiments on visual recognition in the video domain in Sec. 6, significantly improving on the state-of-the-art system for finding people in video sequences [7] both in performance and speed. We also illustrate our approach on the task of action recognition, showing that transfer learning can ameliorate the small-sample problem that plagues current benchmark datasets [18, 19].

## 2   Related Work

Tenenbaum and Freeman [23] introduced bilinear models into the vision community to model data generated from multiple linear factors. Such methods have been extended to the multilinear setting, e.g. by [25], but such models were generally used as a factor analysis or density estimation technique. Recent work has explored extensions of tensor models to discriminant analysis [22, 27], while our work focuses on an efficient max-margin formulation of multilinear models.

There is also a body of related work on learning low-rank matrices from the collaborative filtering literature [21, 17, 16]. Such approaches typically define a convex objective by replacing the $\text{Tr}(W^T W)$ regularization term in our objective (6) with the trace norm $\text{Tr}(\sqrt{W^T W})$. This can be seen as an alternate "soft" rank restriction on $W$ that retains convexity. This is because the trace norm of a matrix is equivalent to the sum of its singular values rather than the number of nonzero eigenvalues (the rank) [3]. Such a formulation would be interesting to pursue in our scenario, but as [17, 16] note, the resulting SDP is difficult to solve. Our approach, though non-convex, leverages existing SVM solvers in the inner loop of a coordinate descent optimization that enforces a hard low-rank condition.

Our bilinear-SVM formulation is closely related to the low-rank SVM formulation of [26]. Wolf et. al. convincingly argue that many forms of visual data are better modeled as matrices rather than vectors - an important motivation for our work (see Fig.1). They analyze the VC dimension of rank constrained linear classifiers and demonstrate an iterative weighting algorithm for approximately solving an SVM problem in which the rank of $W$ acts as a regularizer. They also outline an algorithm similar to the one we propose here which has a hard constraint on the rank, but they include an additional orthogonality constraint on the columns of the factors that compose $W$. This requires cycling through each column separately during the optimization which is presumably slower and may introduce additional local minima. This in turn may explain why they did not present experimental results for their hard-rank formulation.

Our work also stands apart from Wolf et. al. in our focus on the multi-task learning, which dates back at least to the work of Caruna [4]. Our formulation is most similar to that of Ando and Zhang [2]. They describe a procedure for learning linear prediction models for multiple tasks with the assumption that all models share a component living in a common low-dimensional subspace. While this formulation allows for sharing, it does not reduce the number of model parameters as does our approach of sharing factors.

## 3 Model definition

Linear predictors are of the form

$$f_w(x) = w^T x. \tag{1}$$

Existing formulations of linear classification typically treat $x$ as a vector. We argue for many problems, particularly in visual recognition, $x$ is more naturally represented as a matrix or tensor. For example, many state-of-the-art window scanning approaches train a classifier defined over local feature vectors extracted over a spatial neighborhood. The Dalal and Triggs detector [6] is a particularly popular pedestrian detector where $x$ is naturally represented as a concatenation of histogram of gradient (HOG) feature vectors extracted from a spatial grid of $n_y \times n_x$, where each local HOG descriptor is itself composed of $n_f$ features. In this case, it is natural to represent an example $x$ as a tensor $X \in \mathbb{R}^{n_y \times n_x \times n_f}$. For ease of exposition, we develop the mathematics for a simpler matrix representation, fixing $n_f = 1$. This holds, for example, when learning templates defined on grayscale pixel values.

We generalize (1) for a matrix $X$ with

$$f_W(X) = \mathrm{Tr}(W^T X). \tag{2}$$

where both $X$ and $W$ are $n_y \times n_x$ matrices. One advantage of the matrix representation is that it is more natural to regularize $W$ and restrict the number of parameters. For example, one natural mechanism for reducing the degrees of freedom in a matrix is to reduce its rank. We show that one can obtain a biconvex objective function by enforcing a hard restriction on the rank. Specifically, we enforce the rank of $W$ to be at most $d \le \min(n_y, n_x)$. This restriction can be implemented by writing

$$W = W_y W_x^T \qquad \text{where} \qquad W_y \in \mathbb{R}^{n_y \times d}, W_x \in \mathbb{R}^{n_x \times d}. \tag{3}$$

This allows us to write the final predictor explicitly as the following bilinear function:

$$f_{W_y, W_x}(X) = \mathrm{Tr}(W_y W_x^T X) = \mathrm{Tr}(W_y^T X W_x). \tag{4}$$

### 3.1 Learning

Assume we are given a set of training data and label pairs $\{x_n, y_n\}$. We would like to learn a model with low error on the training data. One successful approach is a support vector machine (SVM). We can rewrite the linear SVM formulation for $w$ and $x_n$ with matrices $W$ and $X_n$ using the trace operator.

$$L(w) = \frac{1}{2} w^T w + C \sum_n \max(0, 1 - y_n w^T x_n). \tag{5}$$

$$L(W) = \frac{1}{2} \mathrm{Tr}(W^T W) + C \sum_n \max(0, 1 - y_n \mathrm{Tr}(W^T X_n)). \tag{6}$$

The above formulations are identical when $w$ and $x_n$ are the vectorized elements of matrices $W$ and $X_n$. Note that (6) is convex. We wish to restrict the rank of $W$ to be $d$. Plugging in $W = W_y W_x^T$, we obtain our biconvex objective function:

$$L(W_y, W_x) = \frac{1}{2} \operatorname{Tr}(W_x W_y^T W_y W_x^T) + C \sum_n \max(0, 1 - y_n \operatorname{Tr}(W_x W_y^T X_n)). \qquad (7)$$

In the next section, we show that optimizing (7) over one matrix holding the other fixed is a convex program - specifically, a QP equivalent to a standard SVM. This makes (7) biconvex.

## 3.2 Coordinate descent

We can optimize (7) with a coordinate descent algorithm that solves for one set of parameters holding the other fixed. Each step in this descent is a convex optimization that can be solved with a standard SVM solver. Specifically, consider

$$\min_{W_y} L(W_y, W_x) = \frac{1}{2} \operatorname{Tr}(W_y A W_y^T) + C \sum_n \max(0, 1 - y_n \operatorname{Tr}(W_y^T X_n W_x)). \qquad (8)$$

The above optimization is convex in $W_y$ but does not directly translate into the trace-based SVM formulation from (6). To do so, let us reparametrize $W_y$ as $\tilde{W}_y$:

$$\min_{\tilde{W}_y} L(\tilde{W}_y, W_x) = \frac{1}{2} \operatorname{Tr}(\tilde{W}_y^T \tilde{W}_y) + C \sum_n \max(0, 1 - y_n \operatorname{Tr}(\tilde{W}_y^T \tilde{X}_n)) \qquad (9)$$

where $\quad \tilde{W}_y = W_y A^{\frac{1}{2}} \quad$ and $\quad \tilde{X}_n = X_n W_x A^{-\frac{1}{2}} \quad$ and $\quad A = W_x^T W_x.$

One can see that (9) is structurally equivalent to (6) and hence (5). Hence it can be solved with a standard off-the-shelf SVM solver. Given a solution, we can recover the original parameters by $W_y = \tilde{W}_y A^{-\frac{1}{2}}$. Recall that $A = W_x^T W_x$ is matrix of size $d \times d$ that is in general invertible for small $d$. Using a similar derivation, one can show that $\min_{W_x} L(W_y, W_x)$ is also equivalent to a standard convex SVM formulation.

# 4 Motivation

We outline here a number of motivations for the biconvex objective function defined above.

## 4.1 Regularization

Bilinear models allow a natural way of restricting the number of parameters in a linear model. From this perspective, they are similar to approaches that apply PCA for dimensionality reduction prior to learning. Felzenszwalb et al. [11] find that PCA can reduce the size of HOG features by a factor of 4 without a loss in performance. Image windows are naturally represented as a 3D tensor $X \in \mathbb{R}^{n_y \times n_x \times n_f}$, where $n_f$ is the dimensionality of a HOG feature. Let us "reshape" $X$ into a 2D matrix $X \in \mathbb{R}^{n_{xy} \times n_f}$ where $n_{xy} = n_x n_y$. We can restrict the rank of the corresponding model to $d$ by defining $W = W_{xy} W_f^T$. $W_{xy} \in \mathbb{R}^{n_{xy} \times d}$ is equivalent to a vectorized spatial template defined over $d$ features at each spatial location, while $W_f \in \mathbb{R}^{n_f \times d}$ defines a set of $d$ basis vectors spanning $\mathbb{R}^{n_f}$. This basis can be loosely interpreted as the PCA-basis estimated in [11]. In our biconvex formulation, the basis vectors are not constrained to be orthogonal, but they are learned discriminatively and jointly with the template $W_{xy}$. We show in Sec. 6 this often significantly outperforms PCA-based dimensionality reduction.

## 4.2 Efficiency

Scanning window classifiers are often implemented using convolutions [6, 12]. For example, the product $\operatorname{Tr}(W^T X)$ can be computed for all image windows $X$ with $n_f$ convolutions. By restricting $W$ to be $W_{xy} W_f^T$, we project features into a $d$ dimensional subspace spanned by $W_f$, and compute the final score with $d$ convolutions. One can further improve efficiency by using the same

$d$-dimensional feature space for a large number of different object templates - this is precisely the basis of our transfer approach in Sec.4.3. This can result in significant savings in computation. For example, spatio-temporal templates for finding objects in video tend to have large $n_f$ since multiple features are extracted from each time-slice.

Consider a rank-1 restriction of $W_x$ and $W_y$. This corresponds to a separable filter $W_{xy}$. Hence, our formulation can be used to learn **separable** scanning-window classifiers. Separable filters can be evaluated efficiently with two one-dimensional convolutions. This can result in significant savings because computing the score at the window is now $O(n_x + n_y)$ rather than $O(n_x n_y)$.

### 4.3 Transfer

Assume we wish to train $M$ predictors and are given $\{x_n^m, y_n^m\}$ training data pairs for each prediction problem $1 \le m \le M$. One can write all $M$ learning problems with a single optimization:

$$L(W^1, \ldots, W^M) = \frac{1}{2} \sum_m \mathrm{Tr}(W^{m^T} W^m) + \sum_m C_m \sum_n \max(0, 1 - y_n^m \mathrm{Tr}(W^{m^T} X_n^m)). \quad (10)$$

As written, the problem above can be optimized over each $W^m$ independently. We can introduce a rank constraint on $W^m$ that induces a low-dimensional subspace projection of $X_n^m$. To transfer knowledge between the classification tasks, we require all tasks to use the same low-dimensional subspace projection by sharing the same feature matrix:

$$W^m = W_{xy}^m W_f^T$$

Note that the leading dimension of $W_{xy}^m$ can depend on $m$. This fact allows for $X_n^m$ from different tasks to be of varying sizes. In our motivating application, we can learn a family of HOG templates of varying spatial dimension that share a common HOG feature subspace. The coordinate descent algorithm from Sec.3.2 naturally applies to the multi-task setting. Given a fixed $W_f$, it is straightforward to independently optimize $W_{xy}^m$ by defining $A = W_f^T W_f$. Given a fixed set of $W_{xy}^m$, a single matrix $W_f$ is learned for all classes by computing:

$$\min_{\tilde{W}_f} L(\tilde{W}_f, W_{xy}^1, \ldots, W_{xy}^M) = \frac{1}{2} \mathrm{Tr}(\tilde{W}_f^T \tilde{W}_f) + \sum_m C_m \sum_n \max(0, 1 - y_n^m \mathrm{Tr}(\tilde{W}_f^T \tilde{X}_n^m))$$

$$\text{where} \qquad \tilde{W}_f = W_f A^{\frac{1}{2}} \qquad \text{and} \qquad \tilde{X}_n^m = X_n^m W_{xy}^m A^{-\frac{1}{2}} \qquad \text{and} \qquad A = \sum_m W_{xy}^{m^T} W_{xy}^m.$$

If all problems share the same slack penalty ($C_m = C$), the above can be optimized with an off-the-shelf SVM solver. In the general case, a minor modification is needed to allow for slack-rescaling [24].

In practice, $n_f$ can be large for spatio-temporal features extracted from multiple temporal windows. The above formulation is convenient in that we can use data examples from many classification tasks to learn a good subspace for spatiotemporal features.

## 5 Extensions

### 5.1 Multilinear

In many cases, a data point $x$ is more natural represented as a multidimensional matrix or a high-order tensor. For example, spatio-temporal templates are naturally represented as a $4^{th}$-order tensor capturing the width, height, temporal extent, and the feature dimension of a spatio-temporal window. For ease of exposition let us assume the feature dimension is 1 and so we write a feature vector $x$ as $X \in R^{n_x \times n_y \times n_t}$. We denote the element of a tensor $X$ as $x_{ijk}$. Following [15], we define a scalar product of two tensors $W$ and $X$ as the sum of their element-wise products:

$$\langle W, X \rangle = \sum_{ijk} w_{ijk} x_{ijk}. \quad (11)$$

With the above definition, we can generalize our trace-based objective function (6) to higher-order tensors:

$$L(W) = \frac{1}{2} \langle W, W \rangle + C \sum_n \max(0, 1 - y_n \langle W, X_n \rangle). \quad (12)$$

We wish to impose a rank restriction on the tensor $W$. The notion of rank for tensors of order greater than two is subtle - for example, there are alternate approaches for defining a high-order SVD [25, 15]. For our purposes, we follow [20] and define $W$ as a rank $d$ tensor by writing it as product of matrices $W^y \in \mathbb{R}^{n_y \times d}, W^x \in \mathbb{R}^{n_x \times d}, W^t \in \mathbb{R}^{n_t \times d}$:

$$w_{ijk} = \sum_{s=1}^{d} w_{is}^y w_{js}^x w_{ks}^t. \tag{13}$$

Combining (11) - (13), it is straightforward to show that $L(W^y, W^x, W^t)$ is convex in one matrix given the others. This means our coordinate descent algorithm from Sec.3.2 still applies. As an example, consider the case when $d = 1$. This rank restriction forces the spatio-temporal template $W$ to be separable in along the $x$, $y$, and $t$ axes, allowing for window-scan scoring by three one-dimensional convolutions. This greatly increases run-time efficiency for spatio-temporal templates.

## 5.2 Bilinear structural SVMs

We outline here an extension of our formalism to structural SVMs [24]. Structural SVMs learn models that predict a structured label $y_n$ given a data point $x_n$. Given training data of the form $\{x_n, y_n\}$, the learning problem is:

$$L(w) = \frac{1}{2} w^T w + C \sum_n \max_y (l(y_n, y) - w^T \Delta\phi(x_n, y_n, y)) \tag{14}$$

$$\text{where} \quad \Delta\phi(x_n, y_n, y) = \phi(x_n, y_n) - \phi(x_n, y),$$

and where $l(y_n, y)$ is the loss of assigning example $i$ with label $y$ given that its true label is $y_n$. The above optimization problem is convex in $w$. As a concrete example, consider the task of learning a multiclass SVM for $n_c$ classes using the formalism of Crammer and Singer [5]. Here,

$$w = \begin{bmatrix} w_1^T & \dots & w_{n_c}^T \end{bmatrix},$$

where each $w_i \in \mathbb{R}^{n_x}$ can be interpreted as a classifier for class $i$. The corresponding $\phi(x, y)$ will be a sparse vector with $n_x$ nonzero values at those indices associated with the $y^{th}$ class. It is natural to model the relevant vectors as matrices $W, X_n, \Delta\Phi$ that lie in $\mathbb{R}^{n_c \times n_x}$. We can enforce $W$ to be of rank $d < \min(n_c, n_x)$ by defining $W = W_c W_x^T$ where $W_c \in \mathbb{R}^{n_c \times d}$ and $W_x \in \mathbb{R}^{n_x \times d}$. For example, one may expect template classifiers that classify $n_c$ different human actions to reside in a $d$ dimensional subspace. The resulting biconvex objective function is

$$L(W_c, W_x) = \frac{1}{2} \text{Tr}(W_x W_c^T W_c W_x^T) + C \sum_n \max_y (l(y_n, y) - \text{Tr}(W_x W_c^T \Phi(X_n, y_n, y)). \tag{15}$$

Using our previous arguments, it is straightforward to show that the above objective is biconvex and that each step of the coordinate descent algorithm reduces to a standard structural SVM problem.

## 6  Experiments

We focus our experiments on the task of visual recognition using spatio-temporal templates. This problem domain has large feature sets obtained by histograms of gradients and histograms of optical flow computing from a frame pair. We illustrate our method on two challenging tasks using two benchmark datasets - detecting pedestrians in video sequences from the INRIA-Motion database [7] and classifying human actions in UCF-Sports dataset [18].

We model features computed from frame pairs $x$ as matrices $X \in \mathbb{R}^{n_{xy} \times n_f}$, where $n_{xy} = n_x n_y$ is the vectorized spatial template and $n_f$ is the dimensionality of our combined gradient and flow feature space. We use the histogram of gradient and flow feature set from [7]. Our bilinear model learns a classifier of the form $W_{xy} W_f^T$ where $W_{xy} \in \mathbb{R}^{n_{xy} \times d}$ and $W_f \in \mathbb{R}^{n_f \times d}$. Typical values include $n_y = 14$, $n_x = 6$, $n_f = 84$, and $d = 5$ or $10$.

## 6.1 Spatiotemporal pedestrian detection

**Scoring a detector:** Template classifiers are often scored using missed detections versus false-positives-per-window statistics. However, recent analysis suggests such measurements can be misleading [9]. We opt for the scoring criteria outlined by the widely-acknowledged PASCAL competition [10], which looks at average precision (AP) results obtained after running the detector on cluttered video sequences and suppressing overlapping detections.

**Baseline:** We compare with the linear spatiotemporal-template classifier from [7]. The static-image detector counterpart is a well-known state-of-the-art system for finding pedestrians [6]. Surprisingly, when scoring AP for person detection in the INRIA-motion dataset, we find the spatiotemporal model performed worse than the static-image model. This is corroborated by personal communication with the authors as well as Dalal's thesis [8]. We found that aggressive SVM cutting-plane optimization algorithms [13] were needed for the spatiotemporal model to outperform the spatial model. This suggests our linear baseline is the true state-of-the-art system for finding people in video sequences. We also compare results with an additional rank-reduced baseline obtained by setting $w_f$ to the basis returned by a PCA projection of the feature space from $n_f$ to $d$ dimensions. We use this PCA basis to initialize our coordinate descent algorithm when training our bilinear models.

We show precision-recall curves in Fig.2. We refer the reader to the caption for a detailed analysis, but our bilinear optimization seems to produce the state-of-the-art system for finding people in video sequences, while being an order-of-magnitude faster than previous approaches.

## 6.2 Human action classification

Action classification requires labeling a video sequence with one of $n_c$ action labels. We do this by training $n_c$ 1-vs-all action templates. Template detections from a video sequence are pooled together to output a final action label. We experimented with different voting schemes and found that a second-layer SVM classifier defined over the maximum score (over the entire video) for each template performed well. Our future plan is to integrate the video class directly into the training procedure using our bilinear structural SVM formulation.

Action recognition datasets tend to be quite small and limited. For example, up until recently, the norm consisted of scripted activities on controlled, simplistic backgrounds. We focus our results on the relatively new UCF Sports Action dataset, consisting of non-scripted sequences of cluttered sports videos. Unfortunately, there has been few published results on this dataset, and the initial work [18] uses a slightly different set of classes than those which are available online. The published average class confusion is 69.2%, obtained with leave-one-out cross validation. Using 2-fold cross validation (and hence significantly less training data), our bilinear template achieves a score of 64.8% (Fig. 3). Again, we see a large improvement over linear and PCA-based approaches. While not directly comparable, these results suggest our model is competitive with the state of the art.

**Transfer:** We use the UCF dataset to evaluate transfer-learning in Fig.4. We consider a small-sample scenario when one has only two example video sequences of each action class. Under this scenario, we train one bilinear model in which the feature basis $W_f$ is optimized independently for each action class, and another where the basis is shared across all classes. The independently-trained model tends to overfit to the training data for multiple values of $C$, the slack penalty from (6). The joint model clearly outperforms the independently-trained models.

## 7 Conclusion

We have introduced a generic framework for multilinear classifiers that are efficient to train with existing linear solvers. Multilinear classifiers exploit the natural matrix and/or tensor representation of spatiotemporal data. For example, this allows one to learn separable spatio-temporal templates for finding objects in video. Multilinear classifiers also allow for factors to be shared across classification tasks, providing a novel form of transfer learning. In our future experiments, we wish to demonstrate transfer between domains such as pedestrian detection and action classification.

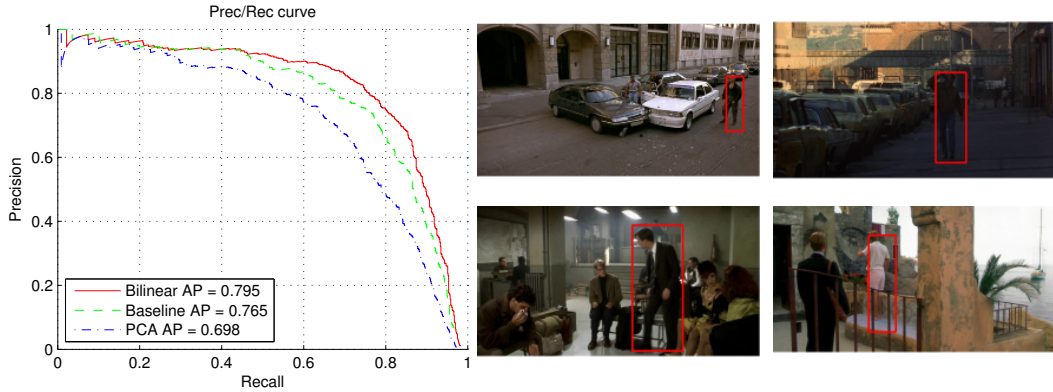

Figure 2: Our results on the INRIA-motion database [7]. We evaluate results using average precision, using the well-established protocol outlined in [10]. The baseline curve is our implementation of the HOG+flow template from [7]. The size of the feature vector is over 7,000 dimensions. Using PCA to reduce the dimensionality by 10X results in a significant performance hit. Using our bilinear formulation with the same low-dimensional restriction, we obtain better performance than the original detector while being 10X faster. We show example detections on video clips on the right.

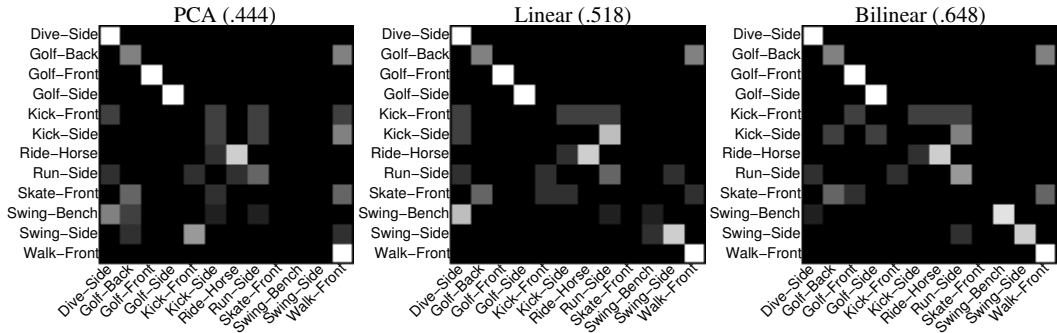

Figure 3: Our results on the UCF Sports Action dataset [18]. We show classification results obtained from 2-fold cross validation. Our bilinear model provides a strong improvement over both the linear and PCA baselines. We show class confusion matrices, where light values correspond to correct classification. We label each matrix with the average classification rate over all classes.

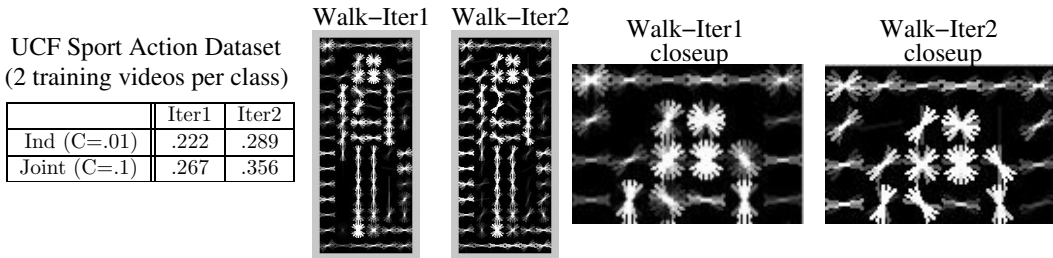

UCF Sport Action Dataset
(2 training videos per class)

|  | Iter1 | Iter2 |
|---|---|---|
| Ind (C=.01) | .222 | .289 |
| Joint (C=.1) | .267 | .356 |

Figure 4: We show results for transfer learning on the UCF action recognition dataset with limited training data - 2 training videos for each of 12 action classes. In the **top table row**, we show results for independently learning a subspace for each action class. In the **bottom table row**, we show results for jointly learning a single subspace that is transfered across classes. In both cases, the regularization parameter $C$ was set on held-out data. The jointly-trained model is able to leverage training data from across all classes to learn the feature space $W_f$, resulting in overall better performance. On the **right**, We show low-rank models $W = W_{xy}W_f^T$ during iterations of the coordinate descent. Note that the head and shoulders of the model are blurred out in iteration 1 which uses PCA, but after the biconvex training procedure discriminatively updates the basis, the final model is sharper at the head and shoulders.

# References

[1] F.A. Al-Khayyal and J.E. Falk. Jointly constrained biconvex programming. *Mathematics of Operations Research*, pages 273–286, 1983.

[2] R.K. Ando and T. Zhang. A framework for learning predictive structures from multiple tasks and unlabeled data. *The Journal of Machine Learning Research*, 6:1817–1853, 2005.

[3] S.P. Boyd and L. Vandenberghe. *Convex optimization*. Cambridge university press, 2004.

[4] R. Caruana. Multitask learning. *Machine Learning*, 28(1):41–75, 1997.

[5] K. Crammer and Y. Singer. On the algorithmic implementation of multiclass kernel-based vector machines. *The Journal of Machine Learning Research*, 2:265–292, 2002.

[6] N. Dalal and B. Triggs. Histograms of oriented gradients for human detection. In *IEEE Computer Society Conference on Computer Vision and Pattern Recognition, 2005. CVPR 2005*, volume 1, 2005.

[7] N. Dalal, B. Triggs, and C. Schmid. Human detection using oriented histograms of flow and appearance. *Lecture Notes in Computer Science*, 3952:428, 2006.

[8] Navneet Dalal. *Finding People in Images and Video*. PhD thesis, Institut National Polytechnique de Grenoble / INRIA Grenoble, July 2006.

[9] P. Dollár, C. Wojek, B. Schiele, and P. Perona. Pedestrian detection: A benchmark. In *CVPR*, June 2009.

[10] M. Everingham, L. Van Gool, C. K. I. Williams, J. Winn, and A. Zisserman. The PASCAL Visual Object Classes Challenge 2008 (VOC2008) Results. http://www.pascal-network.org/challenges/VOC/voc2008/workshop/index.html.

[11] P. Felzenszwalb, R. Girshick, D. McAllester, and D. Ramanan. Object detection with discriminatively trained part based models. *PAMI*, In submission.

[12] P. Felzenszwalb, D. McAllester, and D. Ramanan. A discriminatively trained, multiscale, deformable part model. *Computer Vision and Pattern Recognition, Anchorage, USA, June*, 2008.

[13] V. Franc and S. Sonnenburg. Optimized cutting plane algorithm for support vector machines. In *Proceedings of the 25th international conference on Machine learning*, pages 320–327. ACM New York, NY, USA, 2008.

[14] J. Gorski, F. Pfeuffer, and K. Klamroth. Biconvex sets and optimization with biconvex functions: a survey and extensions. *Mathematical Methods of Operations Research*, 66(3):373–407, 2007.

[15] L.D. Lathauwer, B.D. Moor, and J. Vandewalle. A multilinear singular value decomposition. *SIAM J. Matrix Anal. Appl*, 1995.

[16] N. Loeff and A. Farhadi. Scene Discovery by Matrix Factorization. In *Proceedings of the 10th European Conference on Computer Vision: Part IV*, pages 451–464. Springer-Verlag Berlin, Heidelberg, 2008.

[17] J.D.M. Rennie and N. Srebro. Fast maximum margin matrix factorization for collaborative prediction. In *International Conference on Machine Learning*, volume 22, page 713, 2005.

[18] M.D. Rodriguez, J. Ahmed, and M. Shah. Action MACH a spatio-temporal Maximum Average Correlation Height filter for action recognition. In *IEEE Conference on Computer Vision and Pattern Recognition, 2008. CVPR 2008*, pages 1–8, 2008.

[19] C. Schuldt, I. Laptev, and B. Caputo. Recognizing human actions: A local SVM approach. In *Pattern Recognition, 2004. ICPR 2004. Proceedings of th e17th International Conference on*, volume 3, 2004.

[20] A. Shashua and T. Hazan. Non-negative tensor factorization with applications to statistics and computer vision. In *International Conference on Machine Learning*, volume 22, page 793, 2005.

[21] N. Srebro, J.D.M. Rennie, and T.S. Jaakkola. Maximum-margin matrix factorization. *Advances in Neural Information Processing Systems*, 17:1329–1336, 2005.

[22] D. Tao, X. Li, X. Wu, and S.J. Maybank. General tensor discriminant analysis and Gabor features for gait recognition. *IEEE Transactions on Pattern Analysis and Machine Intelligence*, 29(10):1700, 2007.

[23] J.B. Tenenbaum and W.T. Freeman. Separating style and content with bilinear models. *Neural Computation*, 12(6):1247–1283, 2000.

[24] I. Tsochantaridis, T. Joachims, T. Hofmann, and Y. Altun. Large margin methods for structured and interdependent output variables. *Journal of Machine Learning Research*, 6(2):1453, 2006.

[25] M.A.O. Vasilescu and D. Terzopoulos. Multilinear analysis of image ensembles: Tensorfaces. *Lecture Notes in Computer Science*, pages 447–460, 2002.

[26] L. Wolf, H. Jhuang, and T. Hazan. Modeling appearances with low-rank SVM. In *IEEE Conference on Computer Vision and Pattern Recognition (CVPR)*, pages 1–6. Citeseer, 2007.

[27] S. Yan, D. Xu, Q. Yang, L. Zhang, X. Tang, and H.J. Zhang. Discriminant analysis with tensor representation. In *IEEE Computer Society Conference on Computer Vision and Pattern Recognition*, volume 1, page 526. Citeseer, 2005.

